# AN ARTIFICIAL NEURAL NETWORK FOR SPATIO-TEMPORAL BIPOLAR PATTERNS: APPLICATION TO PHONEME CLASSIFICATION

*Toshiteru Homma*
*Les E. Atlas*
*Robert J. Marks II*

Interactive Systems Design Laboratory
Department of Electrical Engineering, FT-10
University of Washington
Seattle, Washington 98195

## ABSTRACT

An artificial neural network is developed to recognize spatio-temporal bipolar patterns associatively. The function of a formal neuron is generalized by replacing multiplication with convolution, weights with transfer functions, and thresholding with nonlinear transform following adaptation. The Hebbian learning rule and the delta learning rule are generalized accordingly, resulting in the learning of weights and delays. The neural network which was first developed for spatial patterns was thus generalized for spatio-temporal patterns. It was tested using a set of bipolar input patterns derived from speech signals, showing robust classification of 30 model phonemes.

## 1. INTRODUCTION

Learning spatio-temporal (or dynamic) patterns is of prominent importance in biological systems and in artificial neural network systems as well. In biological systems, it relates to such issues as classical and operant conditioning, temporal coordination of sensorimotor systems and temporal reasoning. In artificial systems, it addresses such real-world tasks as robot control, speech recognition, dynamic image processing, moving target detection by sonars or radars, EEG diagnosis, and seismic signal processing.

Most of the processing elements used in neural network models for practical applications have been the formal neuron[1] or its variations. These elements lack a memory flexible to temporal patterns, thus limiting most of the neural network models previously proposed to problems of spatial (or static) patterns. Some past solutions have been to convert the dynamic problems to static ones using buffer (or storage) neurons, or using a layered network with/without feedback.

We propose in this paper to use a "dynamic formal neuron" as a processing element for learning dynamic patterns. The operation of the dynamic neuron is a temporal generalization of the formal neuron. As shown in the paper, the generalization is straightforward when the activation part of neuron operation is expressed in the frequency domain. Many of the existing learning rules for static patterns can be easily generalized for dynamic patterns accordingly. We show some examples of applying these neural networks to classifying 30 model phonemes.

## 2. FORMAL NEURON AND DYNAMIC FORMAL NEURON

The formal neuron is schematically drawn in Fig. 1(a), where

| Input | $\vec{x} = [x_1\, x_2\, \cdots\, x_L]^T$ |
|---|---|
| Activation | $y_i,\ i = 1,2,\ldots,N$ |
| Output | $z_i,\ i = 1,2,\ldots,N$ |
| Transmittance | $\vec{w} = [w_{i1}\, w_{i2}\, \cdots\, w_{iL}]^T$ |
| Node operator | $\eta$ where $\eta(\cdot)$ is a nonlinear memoryless transform |
| Neuron operation | $z_i = \eta(\vec{w}_i^T \vec{x})$        (2.1) |

Note that a threshold can be implicitly included as a transmittance from a constant input.

In its original form of formal neuron, $x_i \in \{0,1\}$ and $\eta(\cdot)$ is a unit step function $u(\cdot)$. A variation of it is a bipolar formal neuron where $x_i \in \{-1,1\}$ and $\eta(\cdot)$ is the sign function $sgn(\cdot)$. When the inputs and output are converted to frequency of spikes, it may be expressed as $x_i \in \mathbf{R}$ and $\eta(\cdot)$ is a rectifying function $r(\cdot)$. Other node operators such as a sigmoidal function may be used.

We generalize the notion of formal neuron so that the input and output are functions of time. In doing so, weights are replaced with transfer functions, multiplication with convolution, and the node operator with a nonlinear transform following adaptation as often observed in biological systems.

Fig. 1(b) shows a schematic diagram of a dynamic formal neuron where

| Input | $\vec{x}(t) = [x_1(t)\, x_2(t)\, \cdots\, x_L(t)]^T$ |
|---|---|
| Activation | $y_i(t),\ i = 1,2,\ldots,N$ |
| Output | $z_i(t),\ i = 1,2,\ldots,N$ |
| Transfer function | $\vec{w}(t) = [w_{i1}(t)\, w_{i2}(t)\, \cdots\, w_{iL}(t)]^T$ |
| Adaptation | $a_i(t)$ |
| Node operator | $\eta$ where $\eta(\cdot)$ is a nonlinear memoryless transform |
| Neuron operation | $z_i(t) = \eta(a_i(-t) \ast \vec{w}_i(t)^T \ast \vec{x}(t))$   (2.2) |

For convenience, we denote $\ast$ as correlation instead of convolution. Note that convolving a(t) with b(t) is equivalent to correlating a(-t) with b(t).

If the Fourier transforms $\vec{x}(f) = F\{\vec{x}(t)\}$, $\vec{w}_i(f) = F\{\vec{w}_i(t)\}$, $y_i(f) = F\{y_i(t)\}$, and $a_i(f) = F\{a_i(t)\}$ exist, then

$$y_i(f) = a_i(f)\,[\vec{w}_i(f)^{CT}\,\vec{x}(f)] \tag{2.3}$$

where $\vec{w}_i(f)^{CT}$ is the conjugate transpose of $\vec{w}_i(t)$.

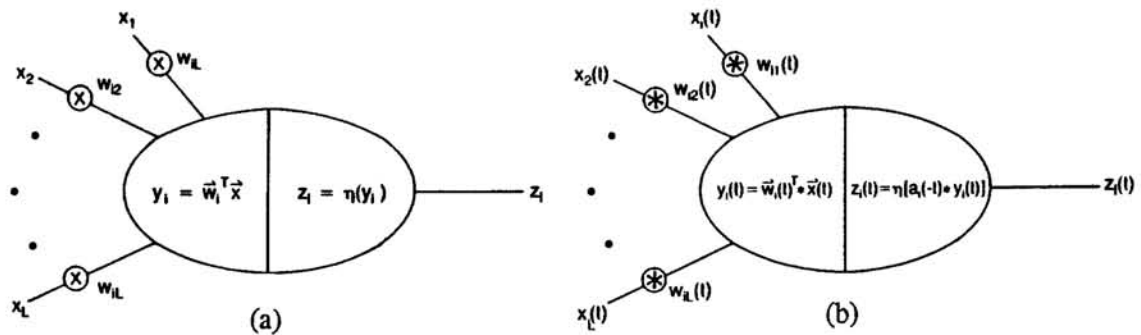

Fig. 1. Formal Neuron and Dynamic Formal Neuron.

## 3. LEARNING FOR FORMAL NEURON AND DYNAMIC FORMAL NEURON

A number of learning rules for formal neurons has been proposed in the past. In the following paragraphs, we formulate a learning problem and describe two of the existing learning rules, namely, Hebbian learning and delta learning, as examples.

Present to the neural network M pairs of input and desired output samples $\{\vec{x}^{(k)}, \vec{d}^{(k)}\}$, $k = 1,2,\ldots,M$, in order. Let $\underline{W}^{(k)} = [\vec{w}_1^{(k)} \; \vec{w}_2^{(k)} \; \cdots \; \vec{w}_N^{(k)}]^T$ where $\vec{w}_i^{(k)}$ is the transmittance vector at the k-th step of learning. Likewise, let

$$\underline{X}^{(k)} = [\vec{x}^{(1)} \; \vec{x}^{(2)} \; \cdots \; \vec{x}^{(k)}], \quad \underline{Y}^{(k)} = [\vec{y}^{(1)} \; \vec{y}^{(2)} \; \cdots \; \vec{y}^{(k)}],$$

$$\underline{Z}^{(k)} = [\vec{z}^{(1)} \; \vec{z}^{(2)} \; \cdots \; \vec{z}^{(k)}], \quad \text{and} \quad \underline{D}^{(k)} = [\vec{d}^{(1)} \; \vec{d}^{(2)} \; \cdots \; \vec{d}^{(k)}],$$

where

$$\vec{y}^{(k)} = \underline{W}^{(k)}\vec{x}^{(k)}, \quad \vec{z}^{(k)} = \underline{\eta}(\vec{y}^{(k)}), \quad \text{and} \quad \underline{\eta}(\vec{y}) = [\eta(y_1) \; \eta(y_2) \; \cdots \; \eta(y_N)]^T.$$

The Hebbian learning rule [2] is described as follows*:

$$\underline{W}^{(k)} = \underline{W}^{(k-1)} + \alpha\vec{d}^{(k)}\vec{x}^{(k)T} \tag{3.1}$$

The delta learning (or LMS learning) rule[3,4] is described as follows:

$$\underline{W}^{(k)} = \underline{W}^{(k-1)} - \alpha\{\underline{W}^{(k-1)}\vec{x}^{(k)} - \vec{d}^{(k)}\}\vec{x}^{(k)T} \tag{3.2}$$

The learning rules described in the previous section are generalized for the dynamic formal neuron by replacing multiplication with correlation. First, the problem is reformulated and then the generalized rules are described as follows.

Present to the neural network M pairs of time-varing input and output samples $\{\vec{x}^{(k)}(t), \vec{d}^{(k)}(t)\}$, $k = 1,2,\ldots,M$, in order. Let $\underline{W}^{(k)}(t) = [\vec{w}_1(t)^{(k)}(t) \; \vec{w}_2^{(k)}(t) \; \cdots \; \vec{w}_N^{(k)}(t)]^T$ where $\vec{w}_i^{(k)}(t)$ is the vector whose elements $w_{ij}^{(k)}(t)$ are transfer functions connecting the input j to the neuron i at the k-th step of learning. The Hebbian learning rule for the dynamic neuron is then

$$\underline{W}^{(k)}(t) = \underline{W}^{(k-1)}(t) + \alpha(-t) * \vec{d}^{(k)}(t) * \vec{x}^{(k)}(t)^T. \tag{3.3}$$

The delta learning rule for dynamic neuron is then

$$\underline{W}^{(k)}(t) = \underline{W}^{(k-1)}(t) - \alpha(-t) * \{\underline{W}^{(k-1)}(t) * \vec{x}^{(k)}(t) - \vec{d}^{(k)}(t)\} * \vec{x}^{(k)}(t)^T. \tag{3.4}$$

This generalization procedure can be applied to other learning rules in some linear discriminant systems[5], the self-organizing mapping system by Kohonen[6], the perceptron[7], the back-propagation model[3], etc. When a system includes a nonlinear operation, more careful analysis is necesssay as pointed out in the Discussion section.

## 4. DELTA LEARNING, PSEUDO INVERSE AND REGULARIZATION

This section reviews the relation of the delta learning rule to the pseudo-inverse and the technique known as regularization.[4,6,8,9,10]

Consider a minimization problem as described below: Find $\underline{W}$ which minimizes

$$R = \sum_k \|\vec{y}^{(k)} - \vec{d}^{(k)}\|_2^2 = (\vec{y}^{(k)} - \vec{d}^{(k)})^T(\vec{y}^{(k)} - \vec{d}^{(k)}) \tag{4.1}$$

subject to $\vec{y}^{(k)} = \underline{W}\vec{x}^{(k)}$.

A solution by the delta rule is, using a gradient descent method,

$$\underline{W}^{(k)} = \underline{W}^{(k-1)} - \alpha\frac{\partial}{\partial \underline{W}}R^{(k)} \tag{4.2}$$

where $R^{(k)} = \| \vec{y}^{(k)} - \vec{d}^{(k)} \|_2^2$ . The minimum norm solution to the problem, $\underline{W}^*$, is unique and can be expressed as

$$\underline{W}^* = \underline{D}\,\underline{X}^t \tag{4.3}$$

where $\underline{X}^t$ is the Moore-Penrose pseudo-inverse of $\underline{X}$ , i.e.,

$$\underline{X}^t = \lim_{\sigma \to 0} (\underline{X}^T \underline{X} + \sigma^2 \underline{I})^{-1} \underline{X}^T = \lim_{\sigma \to 0} \underline{X}^T (\underline{X}\,\underline{X}^T + \sigma^2 \underline{I})^{-1}. \tag{4.4}$$

On the condition that $0 < \alpha < \dfrac{2}{\lambda_{max}}$ where $\lambda_{max}$ is the maximum eigenvalue of $\underline{X}^T\underline{X}$, $\vec{x}^{(k)}$ and $\vec{d}^{(k)}$ are independent, and $\underline{W}^{(k)}$ is uncorrelated with $\vec{x}^{(k)}$,

$$E\{\underline{W}^*\} = E\{\underline{W}^{(\infty)}\} \tag{4.5}$$

where $E\{x\}$ denotes the expected value of $x$. One way to make use of this relation is to calculate $\underline{W}^*$ for known standard data and refine it by (4.2), thereby saving time in the early stage of learning.

However, this solution results in an ill-conditioned $\underline{W}$ often in practice. When the problem is ill-posed as such, the technique known as regularization can alleviate the ill-conditioning of $\underline{W}$ . The problem is reformulated by finding $\underline{W}$ which minimizes

$$R(\sigma) = \sum_k \| \vec{y}^{(k)} - \vec{d}^{(k)} \|_2^2 + \sigma^2 \sum_k \| \vec{w}_k \|_2^2 \tag{4.6}$$

subject to $\vec{y}^{(k)} = \underline{W}\vec{x}^{(k)}$ where $\underline{W} = [\vec{w}_1 \vec{w}_2 \cdots \vec{w}_N]^T$ .
This reformulation regularizes (4.3) to

$$\underline{W}(\sigma) = \underline{D}\,\underline{X}^T(\underline{X}\,\underline{X}^T + \sigma^2\underline{I})^{-1} \tag{4.7}$$

which is statistically equivalent to $\underline{W}^{(\infty)}$ when the input has an additive noise of variance $\sigma^2$ uncorrelated with $\vec{x}^{(k)}$. Interestingly, the leaky LMS algorithm[11] leads to a statistically equivalent solution

$$\underline{W}^{(k)} = \beta\underline{W}^{(k-1)} - \alpha\{\underline{W}^{(k-1)}\vec{x}^{(k)} - \vec{d}^{(k)}\}\vec{x}^{(k)T} \tag{4.8}$$

where $0 < \beta < 1$ and $0 < \alpha < \dfrac{2}{\lambda_{max}}$ . These solutions are related as

$$E\{\underline{W}(\sigma)\} = E\{\underline{W}^{(\infty)}\} \tag{4.9}$$

if $\sigma^2 = \dfrac{1-\beta}{\alpha}$ when $\underline{W}^{(k)}$ is uncorrelated with $\vec{x}^{(k)}$ .[11]

Equation (4.8) can be generalized for a network using dynamic formal neurons, resulting in a equation similar to (3.4). Making use of (4.9), (4.7) can be generalized for a dynamic neuron network as

$$\underline{W}(t\,;\sigma) = F^{-1}\{\underline{D}(f)\underline{X}(f)^{CT}(\underline{X}(f)\underline{X}(f)^{CT} + \sigma^2\underline{I})^{-1}\} \tag{4.10}$$

where $F^{-1}$ denotes the inverse Fourier transform.

## 5. SYNTHESIS OF BIPOLAR PHONEME PATTERNS

This section illustrates the scheme used to synthesize bipolar phoneme patterns and to form prototype and test patterns.

The fundamental and first three formant frequencies, along with their bandwidths, of phonemes provided by Klatt[12] were taken as parameters to synthesize 30 prototype phoneme patterns. The phonemes were labeled as shown in Table 1. An array of L (=100) input neurons covered the range of 100 to 4000 Hz. Each neuron had a bipolar state which was +1 only when one of the frequency bands in the phoneme presented to the network was within the critical band

of the neuron and -1 otherwise. The center frequencies $(f_c)$ of critical bands were obtained by dividing the 100 to 4000 Hz range into a log scale by L. The critical bandwidth was a constant 100 Hz up to the center frequency $f_c = 500$ Hz and $0.2f_c$ Hz when $f_c > 500$ Hz.[13]

The parameters shown in Table 1 were used to construct 30 prototype phoneme patterns. For θ, it was constructed as a combination of t and θ. $F_1$, $F_2$, $F_3$ were the first, second, and third formants, and $B_1$, $B_2$, and $B_3$. were corresponding bandwidths. The fundamental frequency $F_0 = 130$ Hz with $B_0 = 10$ Hz was added when the phoneme was voiced. For plosives, there was a stop before formant traces start. The resulting bipolar patterns are shown in Fig.2. Each pattern had length of 5 time units, composed by linearly interpolating the frequencies when the formant frequency was gliding.

A sequence of phonemes converted from a continuous pronunciation of digits, {o, zero, one, two, three, four, five, six, seven, eight, nine }, was translated into a bipolar pattern, adding two time units of transition between two consequtive phonemes by interpolating the frequency and bandwidth parameters linearly. A flip noise was added to the test pattern and created a noisy test pattern. The sign at every point in the original clean test pattern was flipped with the probability 0.2. These test patterns are shown in Fig. 3.

Table 1. Labels of Phonemes

| Label | Phoneme |
|-------|---------|
| 1 | [iʸ] |
| 2 | [Iᵊ] |
| 3 | [eʸ] |
| 4 | [εᵊ] |
| 5 | [æʳ] |
| 6 | [a] |
| 7 | [ɔᵊ] |
| 8 | [ʌ] |
| 9 | [oʷ] |
| 10 | [ʊᵊ] |
| 11 | [uʷ] |
| 12 | [ɚ] |
| 13 | [aʸ] |
| 14 | [aʷ] |
| 15 | [oʸ] |
| 16 | [w] |
| 17 | [y] |
| 18 | [r] |
| 19 | [l] |
| 20 | [f] |
| 21 | [v] |
| 22 | [θ] |
| 23 | [ð] |
| 24 | [s] |
| 25 | [z] |
| 26 | [p] |
| 27 | [t] |
| 28 | [d] |
| 29 | [k] |
| 30 | [n] |

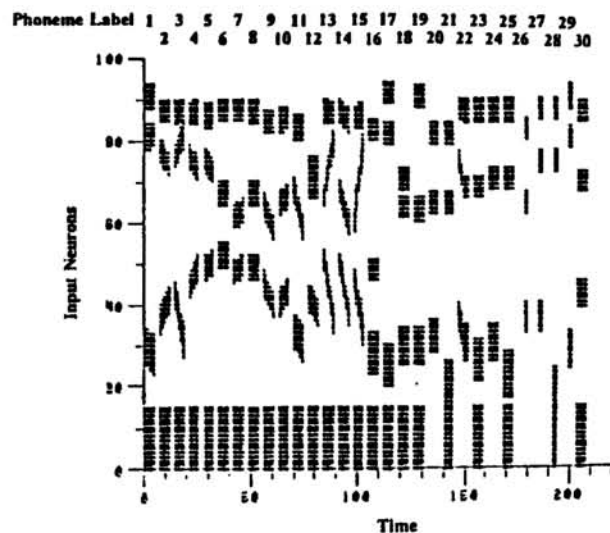

Fig. 2. Prototype Phoneme Patterns. (Thirty phoneme patterns are shown in sequence with intervals of two time units.)

## 6. SIMULATION OF SPATIO-TEMPORAL FILTERS FOR PHONEME CLASSIFICATION

The network system described below was simulated and used to classify the prototype phoneme patterns in the test patterns shown in the previoius section. It is an example of generalizing a scheme developed for static patterns[13] to that for dynamic patterns. Its operation is in two stages. The first stage operation is a spatio-temporal filter bank:

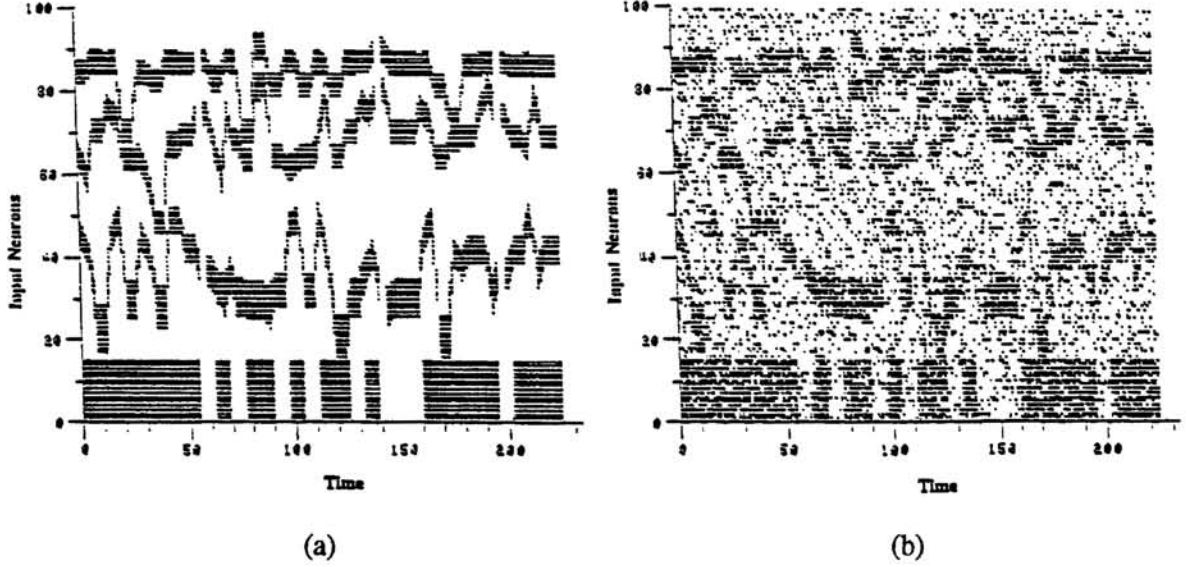

Fig. 3. Test Patterns. (a) Clean Test Pattern. (b) Noisy Test Pattern.

$$\vec{y}(t) = \underline{W}(t) * \vec{x}(t) \text{ , and } \vec{z}(t) = \underline{r}(a(-t)\vec{y}(t)) \text{ .} \tag{6.1}$$

The second stage operation is the "winner-take-all" lateral inhibition:

$$\vec{o}(t) = \vec{z}(t) \text{ , and } \vec{o}(t+\Delta) = \underline{r}(\underline{A}(-t) * \vec{o}(t) - \vec{h}), \tag{6.2}$$

and

$$\underline{A}(t) = (1 + \frac{1}{5N})\underline{I}\delta(t) - \frac{1}{5N}\vec{1}\vec{1}^T \sum_{n=0}^{4}\delta(t-n\Delta). \tag{6.3}$$

where $\vec{h}$ is a constant threshold vector with elements $h_i = h$ and $\delta(\cdot)$ is the Kronecker delta function. This operation is repeated a sufficient number of times, $N_o$.[13,14] The output is $\vec{o}(t + N_o \cdot \Delta)$.

Two models based on different learning rules were simulated with parameters shown below.

**Model 1 (Spatio-temporal Matched Filter Bank)**
Let $\alpha(t) = \delta(t)$ , $\vec{d}^{(k)} = \vec{e}_k$ in (3.3) where $\vec{e}_k$ is a unit vector with its elements $e_{ki} = \delta(k-i)$ .

$$\underline{W}(t) = \underline{X}(t)^T. \tag{6.4}$$

$$h=200, \text{ and } a(t) = \sum_{n=0}^{4}\frac{1}{5}\delta(t-n\Delta).$$

**Model 2 (Spatio-temporal Pseudo-inverse Filter)**
Let $\underline{D} = \underline{I}$ in (4.10). Using the alternative expression in (4.4),

$$\underline{W}(t) = F^{-1}\{(\underline{X}(f)^{CT}\underline{X}(f) + \sigma^2\underline{I})^{-1}X^{CT}\}. \tag{6.5}$$

$$h = 0.05 \text{ , } \sigma^2 = 1000.0 \text{ , and } a(t) = \delta(t).$$

This minimizes

$$R(\sigma,f) = \sum_k \|\vec{y}^{(k)}(f) - \vec{d}^{(k)}(f)\|_2^2 + \sigma^2\sum_k\|\vec{w}_k(f)\|_2^2 \quad \text{for } all \ f \text{ .} \tag{6.6}$$

Because the time and frequency were finite and discrete in simulation, the result of the inverse discrete Fourier transform in (6.5) may be aliased. To alleviate the aliasing, the transfer functions in the prototype matrix $\underline{X}(t)$ were padded with zeros, thereby doubling the lengths. Further zero-padding the transfer functions did not seem to change teh result significantly.

The results are shown in Fig. 4(a)-(d). The arrows indicate the ideal response positions at the end of a phoneme. When the program was run with different thresholds and adaptation function $a(t)$, the result was not very sensitive to the threshold value, but was, nevertheless affected by the choice of the adaptation function. The maximum number of iterations for the lateral inhibition network to converge was observed: for the experiments shown in Fig. 4(a) - (d), the numbers were 44, 69, 29, and 47, respectively. Model 1 missed one phoneme and falsely responded once in the clean test pattern. It missed three and had one false response in the noisy test pattern. Model 2 correctly recognized all phonemes in the clean test pattern, and false-alarmed once in the noisy test pattern.

## 7. DISCUSSION

The notion of convolution or correlation used in the models presented is popular in engineering disciplines and has been applied extensively to designing filters, control systems, etc. Such operations also occur in biological systems and have been applied to modeling neural networks.[15,16] Thus the concept of dynamic formal neuron may be helpful for the improvement of artificial neural network models as well as the understanding of biological systems. A portion of the system described by Tank and Hopfield [17] is similar to the matched filter bank model simulated in this paper.

The matched filter bank model (Model 1) performs well when all phonemes (as above) are of the same duration. Otherwise, it would perform poorly unless the lengths were forced to a maximum length by padding the input and transfer functions with -1's during calculation. The pseudo-inverse filter model, on the other hand, should not suffer from this problem. However, this aspect of the model (Model 2) has not yet been explicitly simulated.

Given a spatio-temporal pattern of size $L \times K$, i.e., L spatial elements and K temporal elements, the number of calculations required to process the first stage of filtering by both models is the same as that by a static formal neuron network in which each neuron is connected to the $L \times K$ input elements. In both cases, $L \times K$ multiplications and additions are necessary to calculate one output value. In the case of bipolar patterns, the mutiplication used for calculation of activation can be replaced by sign-bit check and addition. A future investigation is to use recursive filters or analog filters as transfer functions for faster and more efficient calculation. There are various schemes to obtain optimal recursive or analog filters.[18,19] Besides the lateral inhibition scheme used in the models, there are a number of alternative procedures to realize a "winner-take-all" network in analog or digital fashion.[15,20,21]

As pointed out in the previous section, the Fourier transform in (6.5) requires a precaution concerning the resulting length of transfer functions. Calculating the recursive correlation equation (3.4) also needs such preprocessing as windowing or truncation.[22]

The generalization of static neural networks to dynamic ones along with their learning rules is straightforward as shown if the neuron operation and the learning rule are linear. Generalizing a system whose neuron operation and/or learning rule are nonlinear requires more careful analysis and remains for future work. The system described by Watrous and Shastri[16] is an example of generalizing a backpropagation model. Their result showed a good potential of the model and a need for more rigorous analysis of the model. Generalizing a system with recurrent connections is another task to be pursued. In a system with a certain analytical nonlinearity, the signals are expressed by Volterra functionals, for example. A practical learning system can then be constructed if higher kernels are neglected. For example, a cubic function can be used instead of a sigmoidal function.

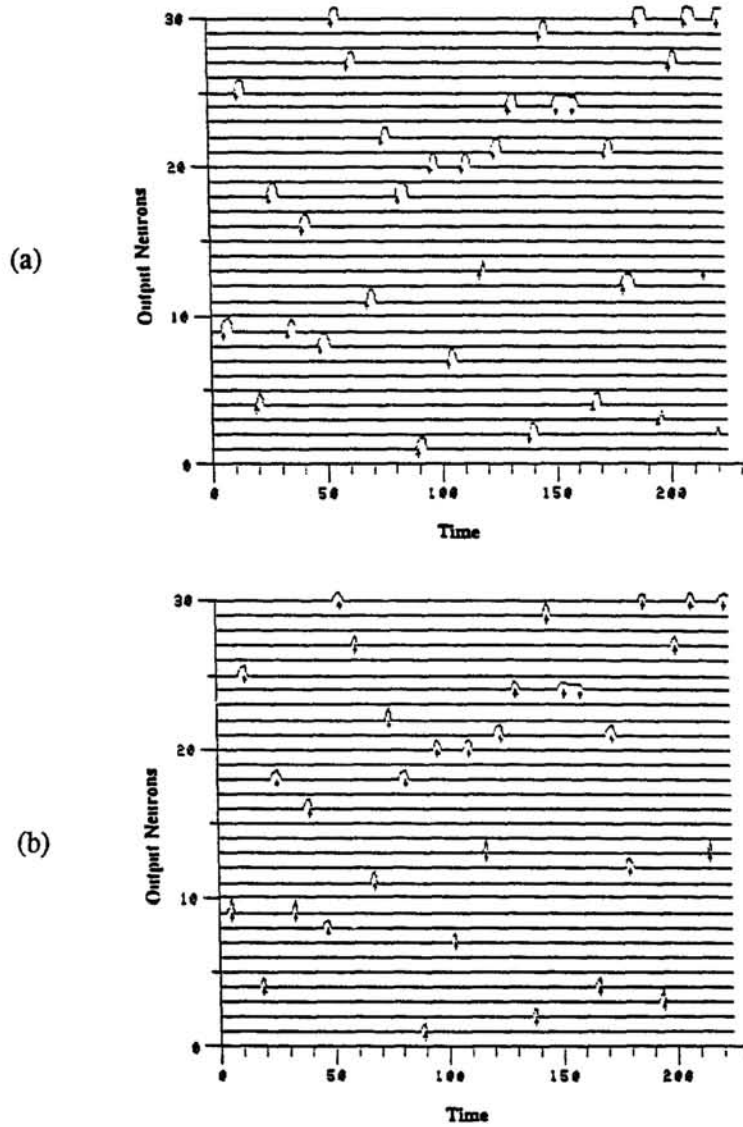

Fig. 4. Performance of Models. (a) Model 1 with Clean Test Pattern. (b) Model 2 with Clean Test Pattern. (c) Model 1 with Noisy Test Pattern. (d) Model 2 with Noisy Test Pattern. Arrows indicate the ideal response positions at the end of phoneme.

## 8. CONCLUSION

The formal neuron was generalized to the dynamic formal neuron to recognize spatio-temporal patterns. It is shown that existing learning rules can be generalized for dynamic formal neurons.

An artificial neural network using dynamic formal neurons was applied to classifying 30 model phonemes with bipolar patterns created by using parameters of formant frequencies and their bandwidths. The model operates in two stages: in the first stage, it calculates the correlation between the input and prototype patterns stored in the transfer function matrix, and, in the second stage, a lateral inhibition network selects the output of the phoneme pattern close to the input pattern.

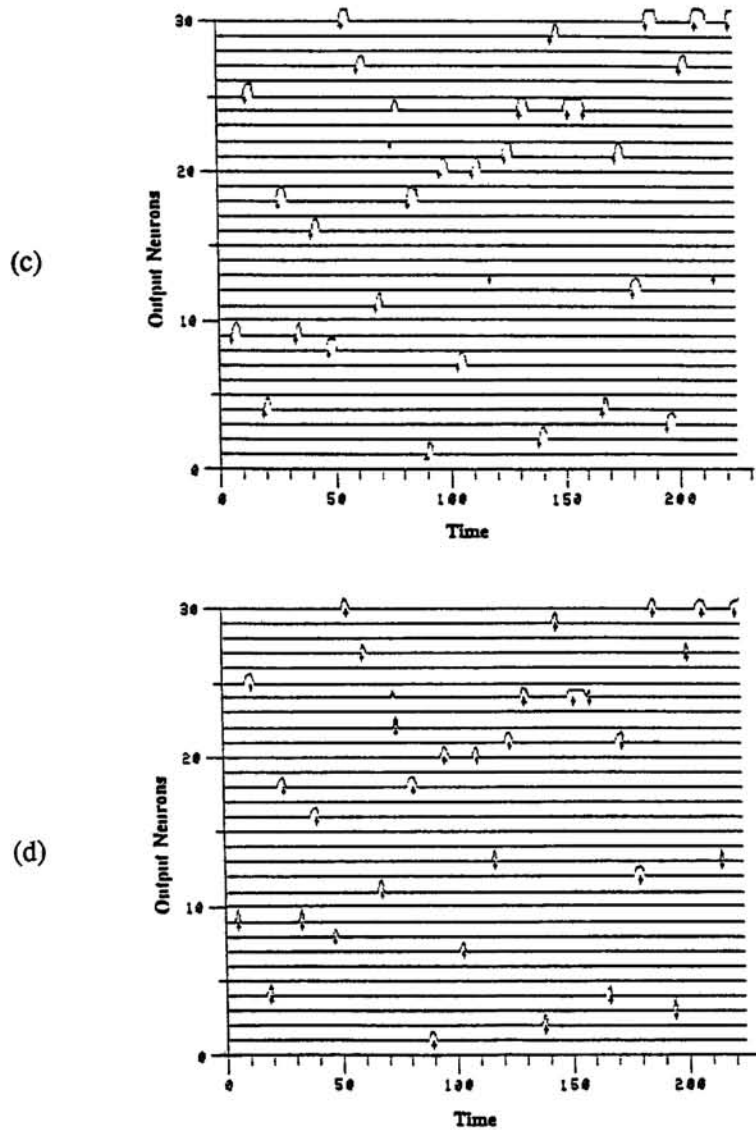

(c)

(d)

Fig. 4 (continued.)

Two models with different transfer functions were tested. Model 1 was a matched filter bank model and Model 2 was a pseudo-inverse filter model. A sequence of phoneme patterns corresponding to continuous pronunciation of digits was used as a test pattern. For the test pattern, Model 1 missed to recognize one phoneme and responded falsely once while Model 2 correctly recognized all the 32 phonemes in the test pattern. When the flip noise which flips the sign of the pattern with the probability 0.2, Model 1 missed three phonemes and falsely responded once while Model 2 recognized all the phonemes and false-alarmed once. Both models detected the phonems at the correct position within the continuous stream.

## Footnotes

* This interpretation assumes a strong supervising signal at the output while learning.

### References

1. W. S. McCulloch and W. Pitts, "A logical calculus of the ideas imminent in nervous activity," *Bulletin of Mathematical Biophysics*, vol. 5, pp. 115-133, 1943.

2. D. O. Hebb, *The Organization of Behavior*, Wiley, New York, 1949.

3.    D. E. Rumelhart, G. E. Hinton, and R. J. Williams, "Learning internal representations by error propagation," in *Parallel Distributed Processing, Vol. 1*, MIT, Cambridge, 1986.

4.    B. Widrow and M. E. Hoff, "Adaptive switching circuits," *Institute of Radio Engineers, Western Electronics Show and Convention*, vol. Convention Record Part 4, pp. 96-104, 1960.

5.    R. O. Duda and P. E. Hart, *Pattern Classification and Scene Analysis*, Chapter 5, Wiley, New York, 1973.

6.    T. Kohonen, *Self-organization and Associative Memory*, Springer-Verlag, Berlin, 1984.

7.    F. Rosenblatt, *Principles of Neurodynamics*, Spartan Books, Washington, 1962.

8.    J. M. Varah, "A practical examination of some numerical methods for linear discrete ill-posed problems," *SIAM Review*, vol. 21, no. 1, pp. 100-111, 1979.

9.    C. Koch, J. Marroquin, and A. Yuille, "Analog neural networks in early vision," *Proceedings of the National Academy of Sciences, USA*, vol. 83, pp. 4263-4267, 1986.

10.    G. O. Stone, "An analysis of the delta rule and the learning of statistical associations," in *Parallel Distributed Processing., Vol. 1*, MIT, Cambridge, 1986.

11.    B. Widrow and S. D. Stearns, *Adaptive Signal Processing*, Prentice-Hall, Englewood Cliffs, 1985.

12.    D. H. Klatt, "Software for a cascade/parallel formant synthesizer," *Journal of Acoustical Society of America*, vol. 67, no. 3, pp. 971-995, 1980.

13.    L. E. Atlas, T. Homma, and R. J. Marks II, "A neural network for vowel classification," *Proceedings International Conference on Acoustics, Speech, and Signal Processing*, 1987.

14.    R. P. Lippman, "An introduction to computing with neural nets," *IEEE ASSP Magazine*, April, 1987.

15.    S. Amari and M. A. Arbib, "Competition and cooperation in neural nets," in *Systems Neuroscience*, ed. J. Metzler, pp. 119-165, Academic Press, New York, 1977.

16.    R. L. Watrous and L. Shastri, "Learning acoustic features from speech data using connectionist networks," *Proceedings of The Ninth Annual Conference of The Cognitive Science Society*, pp. 518-530, 1987.

17.    D. Tank and J. J. Hopfield, "Concentrating information in time: analog neural networks with applications to speech recognition problems," *Proceedings of International Conference on Neural Netoworks*, San Diego, 1987.

18.    J. R. Treichler, C. R. Johnson, Jr., and M. G. Larimore, *Theory and Design of Adaptive Filters*, Chapter 5, Wiley, New York, 1987.

19.    M Schetzen, *The Volterra and Wiener Theories of Nonlinear Systems*, Chapter 16, Wiley, New York, 1980.

20.    S. Grossberg, "Associative and competitive principles of learning," in *Competition and Cooperation in Neural Nets*, ed. M. A. Arbib, pp. 295-341, Springer-Verlag, New York, 1982.

21.    R. J. Marks II, L. E. Atlas, J. J. Choi, S. Oh, K. F. Cheung, and D. C. Park, "A performance analysis of associative memories with nonlinearities in the correlation domain," (submitted to Applied Optics), 1987.

22.    D. E. Dudgeon and R. M. Mersereau, *Multidimensional Digital Signal Processing*, pp. 230-234, Prentice-Hall, Englewood Cliffs, 1984.
